# Ultrafast classical phylogenetic method beats large protein language models on variant effect prediction

**Sebastian Prillo**[*]
University of California, Berkeley
sprillo@berkeley.edu

**Wilson Wu**[*]
University of California, Berkeley
nosliw@berkeley.edu

**Yun S. Song**
University of California, Berkeley
yss@berkeley.edu

## Abstract

Amino acid substitution rate matrices are fundamental to statistical phylogenetics and evolutionary biology. Estimating them typically requires reconstructed trees for massive amounts of aligned proteins, which poses a major computational bottleneck. In this paper, we develop a near-linear time method to estimate these rate matrices from multiple sequence alignments (MSAs) alone, thereby speeding up computation by orders of magnitude. Our method relies on a near-linear time cherry reconstruction algorithm which we call *FastCherries* and it can be easily applied to MSAs with millions of sequences. On both simulated and real data, we demonstrate the speed and accuracy of our method as applied to the classical model of protein evolution. By leveraging the unprecedented scalability of our method, we develop a new, rich phylogenetic model called *SiteRM*, which can estimate a general *site-specific* rate matrix for each column of an MSA. Remarkably, in variant effect prediction for both clinical and deep mutational scanning data in ProteinGym, we show that despite being an independent-sites model, our SiteRM model outperforms large protein language models that learn complex residue-residue interactions between different sites. We attribute our increased performance to conceptual advances in our probabilistic treatment of evolutionary data and our ability to handle extremely large MSAs. We anticipate that our work will have a lasting impact across both statistical phylogenetics and computational variant effect prediction. FastCherries and SiteRM are implemented in the CherryML package https://github.com/songlab-cal/CherryML.

## 1 Introduction

Amino acid substitution rate matrices are crucial to statistical phylogenetics and evolutionary biology. Perhaps best known for their use in phylogenetic tree reconstruction [1, 2, 3, 4, 5], these transition-rate matrices allow us to infer the evolutionary history of a protein family. This has given insights into protein evolution, function, and fitness, which among other important applications, has informed the development of vaccines during the COVID-19 pandemic [6]. Statistically speaking, transition-rate matrices are *parameters* of continuous-time Markov chain models describing protein evolution. One of the most popular such models to date is the seminal model of Le and Gascuel [7] (LG for short), which posits that each site of a protein evolves independently following a continuous-time Markov chain parameterized by a global transition-rate matrix $Q \in \mathbb{R}^{20 \times 20}$ up to site-specific rates.

---

[*]Equal contribution; authors listed alphabetically

Unfortunately, estimating transition-rate matrices under the LG and related classical models from a set of multiple sequence alignments (MSAs) is a challenging task. Indeed, one has to first estimate phylogenetic trees and site-specific rates for each MSA, and then estimate the transition-rate matrix (or matrices) $Q$ which best explain the data. The procedures of tree plus site-rate estimation and of transition-rate matrix estimation are usually performed in a coordinate-ascent fashion until convergence. Both of these steps are computationally challenging: maximum likelihood estimation (MLE) of trees is in general NP-hard [8], while MLE of a rate matrix $Q$ given MSAs, trees and site-rates has historically proven a computationally demanding task [9, 10].

Fortunately, recent work has made it possible to perform MLE of transition-rate matrices given MSAs, trees and site-rates in a scalable fashion. The method, called CherryML [11], replaces the full joint likelihood of the MSA data with a *composite* likelihood over cherries in the trees. When applied to the LG model, CherryML is orders of magnitude faster than traditional MLE with the Expectation-Maximization (EM), all while sacrificing a relatively small amount of statistical efficiency, estimated to be around $50\%$ on simulation studies. The method was also shown to perform on-par with EM on real datasets when evaluated on held-out likelihood. While the work of CherryML represents a significant advance, the method still requires expensive phylogenetic tree reconstruction as a prerequisite step. This hinders the scalability of the method when applied to many MSAs, some with millions of sequences. In fact, in the original CherryML work, MSAs were subsampled down to a thousand sequences each to alleviate the burden of tree reconstruction.

In this work, we introduce a new methodology to significantly speed up the end-to-end estimation of rate matrices under the LG model, and extend this methodology to estimate site-specific rate matrices. Specifically, we speed up the phylogenetic tree plus site-rate estimation step required by the CherryML method, using a method which we call *FastCherries*. The resulting end-to-end method consisting of CherryML with FastCherries can, given a set of starting MSAs, estimate site-specific rates and an accurate transition-rate matrix in *nearly linear* time in the input dataset size. The method is thus essentially computationally optimal up to logarithmic factors and other constants. On a real MSA with more than 450,000 sequences of length 364, FastCherries took just 1,000 seconds on 1 CPU core, and we can estimate site-specific rate matrices at an additional cost of *one second* per site.

Thanks to its unprecedented scalability, we applied CherryML with FastCherries to estimate *site-specific transition-rate matrices* for MSAs with hundreds of thousands of sequences. This model, which we call *SiteRM* for short, posits that each site $i$ of a protein family $f$ evolves independently according to a family- and site-specific rate matrix $Q_i^f$. Strikingly, we show that this *independent-sites* model excels at variant effect prediction, outperforming the seminal EVmutation [12] model *with epistatic interactions* as well as many large protein language models such as an ESM-1v ensemble [13]. We attribute the increased performance of SiteRM to the principled probabilistic treatment of the evolutionary process, which is absent in competing approaches. This opens up a new avenue to variant effect prediction via the use of probabilistic models of protein evolution in time. FastCherries and SiteRM are available at `https://github.com/songlab-cal/CherryML`.

## 2 Background and Related Work

### 2.1 Classical models of protein evolution

As time passes, proteins experience mutations, many of which are neutral, having no effect on fitness. On the other hand, some mutations are deleterious, leading to decreased fitness and hence negative selection, while other mutations confer increased fitness and thus are favored by natural selection. This complex evolutionary process of mutation and selection leads to the vast biological diversity observed today, and modeling it is the goal of statistical phylogenetics. Formally, a model of protein evolution is a conditional distribution $p(y|x, t, f)$ describing the probability of sequence $y$, the evolved from sequence $x$ after time $t$, in protein family $f$.

**Standard probabilistic models of protein evolution.** Early work on protein evolution relied on counting-based heuristics, such as the seminal works of Dayhoff et al. [14] and JTT [15]. The first probabilistic model of protein evolution was proposed by Whelan and Goldman [16]. It posits that each site of any protein evolves i.i.d. following a time-reversible continuous-time Markov chain parameterized by a global transition-rate matrix $Q \in \mathbb{R}^{20 \times 20}$, normalized so that the expected number of substitutions in a unit of time equals 1. Each protein family $f$ may evolve at a different rate $\alpha_f$. Letting $l_f$ be the number of columns of an MSA for protein family $f$, we have $p_{\text{WAG}}(y|x, t, f) =$

$\prod_{i=1}^{l_f} \exp(\alpha_f tQ)[x_i, y_i]$, where $\exp(M)[a, b]$ denotes the $(a, b)$ entry of the matrix exponential $\exp(M)$. The seminal extension of the above model by Le and Gascuel [7] still uses a global, normalized transition-rate matrix $Q$, but incorporates site-specific rates $(\alpha_1^f, \alpha_2^f, \ldots, \alpha_{l_f}^f)$, giving

$$p_{\text{LG}}(y|x, t, f) = \prod_{i=1}^{l_f} \exp(\alpha_i^f tQ)[x_i, y_i].$$

This is one of the most popular models of protein evolution to date, used extensively in phylogenetic tree reconstruction. The LG model has a total of $400 + \sum_{f=1}^{m} l_f$ parameters, where $m$ denotes the number of protein families under study. The site-specific rates $\alpha_i^f$ are typically constrained with a model of site-rate variation, such as the $\Gamma$ model [17] or the probability-distribution-free model.

**Phylogenetic models.** Unfortunately, estimating models of protein evolution is a challenging task. Indeed, one does not have access to training data of the form $(x, y, t, f)$. Instead, one has access to MSAs. Formally, let $D = (D_1, \ldots, D_m)$ be MSAs for $m$ protein families. The data $D$ are modeled with a *phylogenetic model* [11], which is parameterized by (1) a model of protein evolution $p(y|x, t, f)$ (as described above), (2) phylogenetic trees $\mathcal{T}_1, \ldots, \mathcal{T}_m$ (one tree per MSA), and (3) a root state distribution $\pi_{\text{root}}(x)$. The phylogenetic model posits that the MSA data were generated by sampling the root state from each tree following $\pi_{\text{root}}(x)$ and then running the evolutionary model $p(y|x, t, f)$ down each tree. We denote the phylogenetic model's likelihood as $p_{\text{phylo}}(D|\mathcal{T})$. To estimate a model of protein evolution $p(y|x, t, f)$ with MLE from data from a phylogenetic model $p_{\text{phylo}}(D|\mathcal{T})$, one thus has to deal with the nuisance parameters $\mathcal{T} = (\mathcal{T}_1, \mathcal{T}_2, \ldots, \mathcal{T}_m)$. Typically, this is done with coordinate ascent, wherein one alternately optimizes the trees $\mathcal{T}$ plus site-rates $\alpha_i^f$ given the transition-rate matrix $Q$ and then optimizes the transition-rate matrix $Q$ given the tree estimates $\mathcal{T}$ plus site-rates. Both of these steps have historically proven to be computationally demanding.

## 2.2   CherryML

Recently, the work of CherryML [11] introduced a scalable and accurate method for estimating the transition-rate matrix (or matrices) $Q$ given the trees $\mathcal{T}$ and site-rates $\alpha_i^f$. It is assumed, as is typical in statistical phylogenetics, that $Q$ is time-reversible. For the LG model, this method was shown to be orders of magnitude faster than the traditional EM method. In fact, CherryML's runtime for the LG model is linear in the input data size up to logarithmic factors, making it essentially computationally optimal. From a statistical point of view, CherryML is just $\sim 50\%$ less efficient than full MLE.

**Composite likelihoood.**   The key to CherryML lies in the use of a composite likelihood, which generally leads to consistent parameter estimates under weak assumptions [18]. Adapting the notation of CherryML [11], for a generic phylogenetic model $p_{\text{phylo}}$, the full joint likelihood of the data is $\ell = \log p_{\text{phylo}}(D|\mathcal{T}) = \sum_{f=1}^{m} \log p_{\text{phylo}}(D_f|\mathcal{T}_f)$. CherryML replaces the MSA's log-likelihood $\log p_{\text{phylo}}(D_f|\mathcal{T}_f)$ by a composite likelihood over cherries in the trees, where cherries are iteratively picked until either 0 or 1 leaf remains. Specifically, letting $\{(u_j^f, v_j^f)\}_{1 \leq j \leq c_f}$ be the $c_f$ cherries in tree $\mathcal{T}_f$, CherryML considers the composite likelihood $\ell_{\text{comp}} = \sum_{f=1}^{m} \sum_{j=1}^{c_f} \log p_{\text{phylo}}(D_f[u_j^f] \mid D_f[v_j^f], \mathcal{T}_f)$, where $D_f[u_j^f]$ denotes the sequence in MSA $D_f$ corresponding to leaf $u_j^f$. For a stationary time-reversible model, the term $p_{\text{phylo}}(D_f[u_j^f]|D_f[v_j^f], \mathcal{T}_f)$ is exactly equal to $p(D_f[u_j^f]|D_f[v_j^f], t_j^f)$ where $t_j^f$ is the distance between $u_j^f$ and $v_j^f$ in tree $\mathcal{T}_f$. Therefore, the composite likelihood reduces to $\ell_{\text{comp}} = \sum_{f=1}^{m} \sum_{j=1}^{c_f} \log p(D_f[u_j^f]|D_f[v_j^f], t_j^f, f)$. This way, CherryML reduces the problem of learning the transition-rate matrix (or matrices) $Q$ given the trees $\mathcal{T}$, site-rates, and MSAs $D$ to a supervised learning problem over the data $(x, y, t, f)$ given by $(D_f[u_j^f], D_f[v_j^f], t_j^f, f)$. Cherries are considered in both directions when forming the composite likelihood, so that if $(x, y, t, f)$ is a training datapoint, then so is $(y, x, t, f)$. Thus, CherryML can be viewed as a modern, principled version of the JTT method [15] where sequences are paired and then rate matrix estimation proceeds by MLE rather than via counting heuristics.

**Time quantization.** CherryML further simplifies the composite likelihood by quantizing (or discretizing) time into a finite number $b$ of values $\tau_1 < \tau_2 < \cdots < \tau_b$. Letting $q(t)$ be the quantized value of $t$ (which is chosen to minimize relative error), in the LG model with site-rates $r_k^f$ for site $1 \leq k \leq l_f$ of family $f$, CherryML leads to the quantized composite likelihood $\ell_{\text{comp, quant}} = \sum_{f=1}^{m} \sum_{j=1}^{c_f} \sum_{i=1}^{l_f} \log(\exp(q(r_i^f t_j^f)Q)[D_f[u_j^f]_i, D_f[v_j^f]_i])$. The terms in this expres-

sion can be grouped together by quantization time $\tau_k$, leading to

$$\ell_{\text{comp, quant}} = \sum_{k=1}^{b} \langle C_k, \log \exp(\tau_k Q) \rangle,$$

where $C_k$ is a $20 \times 20$ count matrix for the number of transitions between two amino acids at a quantized distance of $\tau_k$, and $\langle \cdot, \cdot \rangle$ denotes the matrix inner product. This function can be evaluated in time $\mathcal{O}(bs^3)$ where $s = 20$ is the number of states, which is remarkable since this no longer depends on the input dataset size. It can be optimized using automatic differentiation software such as PyTorch [19] and Tensorflow [20]. CherryML leverages a differentiable implementation of the matrix exponential operator based on robust Taylor expansions [21] and utilizes the Adam optimizer [22].

## 3 Method

### 3.1 The SiteRM model

In this work, we propose a richer model of protein evolution, which we call the *SiteRM* model:

$$p_{\text{SiteRM}}(y|x, t, f) = \prod_{i=1}^{l_f} \exp(tQ_i^f)[x_i, y_i],$$

where $Q_1^f, \ldots, Q_{l_f}^f$ are site-specific transition-rate matrices. Unlike $Q$-matrix described above, $Q_i^f$ are *not* normalized and they subsume site-specific rates. This model has $400 \times \sum_{f=1}^{m} l_f$ parameters and strictly generalizes the LG model. This is a particular case of the 'partition model' available in IQTree [5], in which each site forms its own group in the partition.

### 3.2 Near-linear time end-to-end estimation of model parameters

Here, we present an algorithm for speeding up end-to-end estimation of model parameters. Related work on speeding up the tree reconstruction step required for rate matrix estimation is described in Appendix A.1. Although we focus on the LG model [7] for the sake of analyzing statistical efficiency, our method will also enable fitting site-specific rate matrices under the richly parameterized SiteRM model, as we illustrate on the task of variant effect prediction. For the LG model, our work speeds up tree and site-rate estimation given the rate matrix $Q$ and MSAs $D$. Together with CherryML's procedure for estimating a transition-rate matrix $Q$ given trees, site-rates, and MSAs, this leads to a full coordinate-ascent procedure with near-linear time updates, making it exceptionally scalable.

**Method overview.** The key idea of our method is that CherryML's composite likelihood depends on the trees $\mathcal{T}$ only through the cherries $(u_j^f, v_j^f)$ and their quantized distances $t_j^f$. In other words, there is no point in estimating the full trees $\mathcal{T}$ if only their cherries will be used. Our method thus proceeds by first grouping the sequences within each MSA $D_f$ into disjoint pairs using a near-linear time divide-and-conquer approach which tries to minimize total Hamming distance across all pairs. Letting $n_f$ be the number of sequences in MSA $D_f$ (so that MSA $D_f$ has size $n_f \times l_f$), we obtain $c_f = \lfloor n_f/2 \rfloor$ pairs $(u_1^f, v_1^f), \ldots, (u_{c_f}^f, v_{c_f}^f)$. These pairs are the putative cherries of the tree $\mathcal{T}_f$. Next, the site-specific rates $\alpha_1^f, \ldots, \alpha_{l_f}^f$ and the quantized times $t_1^f, \ldots, t_{c_f}^f$ separating each pair of sequences are estimated using coordinate ascent. We call our method *FastCherries*. A schematic of the FastCherries/SiteRM method is provided in Figure 1.

**Divide-and-conquer.** To pair up the set of sequences $S$, we use an almost-linear time distance-based tree topology reconstruction algorithm based on divide-and-conquer. The general operation of the algorithm is as follows. Let $d$ be a dissimilarity function between protein sequences. We use the normalized Hamming distance $d$ ignoring gaps, but other dissimilarities may be used. Assuming $|S| \geq 3$, we first try to find the diameter of the set $S$ (i.e., the two furthest away sequences) as follows: (1) take a random sequence $z_0 \in S$, (2) find the furthest sequence to $z_0$ in $S$: $z_1 = \arg\max_{z \in S} d(z, z_0)$; (3) find the furthest sequences to $z_1$ in $S$: $z_2 = \arg\max_{z \in S} d(z, z_1)$. Our putative diametrically opposite sequences are given by $z_1$ and $z_2$. Note that this is analogous to the well-known linear-time algorithm used to find the diameter of a tree. We now use $z_1$ and $z_2$ as pivots to split the set $S$ into two subsets $S_1$ and $S_2$ based on distance to $z_1$ and $z_2$:

$$S_1 = \{z \in S : d(z, z_1) \leq d(z, z_2)\} \quad \text{and} \quad S_2 = \{z \in S : d(z, z_1) > d(z, z_2)\}.$$

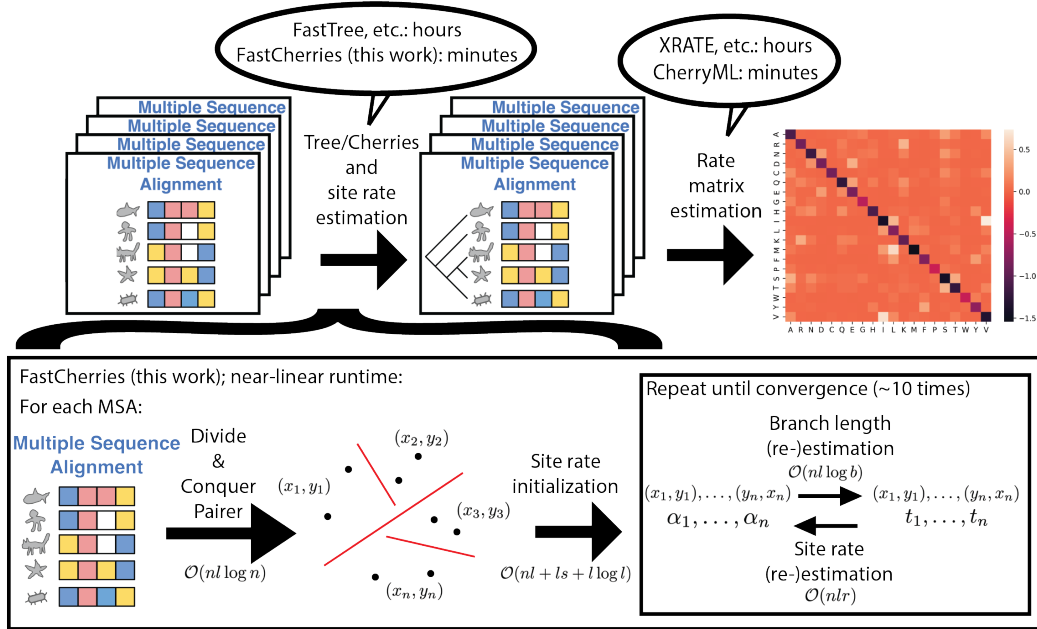

Figure 1: **A schematic illustration of our FastCherries/SiteRM method.** Rate matrix estimation from a set of MSAs classically proceeds in two steps: tree estimation, followed by rate matrix estimation. The recently proposed CherryML method [11] significantly speeds up the rate matrix estimation step. Since CherryML only requires the cherries in the trees, we propose FastCherries, a new near-linear method that estimates only the cherries in the tree (as well as the site rates) rather than the whole tree. FastCherries proceeds in two steps: a divide-and-conquer pairing step based on Hamming distance, followed by site rate and branch length estimation. Site rate and branch length estimation alternate until convergence. CherryML's speed allows estimating not only a single global rate matrix, but also *one rate matrix per site*, which we call the SiteRM model. In this schematic, computational complexities for each step of FastCherries is annotated at each step; $n$ = number of sequences in the MSA, $l$ = number of sites in the MSA, $s$ = number of states (e.g., 20 for amino acids), $r$ = number of site rate categories of the LG model (e.g., 4 or 20 is typical), $b$ = number of quantization points used to quantize time by CherryML. Precomputation of the matrix exponentials — which is shared across all MSAs — is excluded from the schematic and costs $\mathcal{O}(rbs^3)$. MSA illustrations adapted from [23].

We recurse the above procedure on $S_1$ and $S_2$. The base case consists of a set $S$ with either 1 or 2 sequences. If $S$ has 2 sequences, we pair them. Otherwise, we return the lone sequence, which will get paired with another lone sequence up in the recursive calls if both $|S_1|$ and $|S_2|$ are odd. This way, all sequences except at most one (when $|S|$ is odd) will be paired together. Assuming balanced splits of size $0 < \epsilon < |S_1|/|S_2| < 1/\epsilon$ for some universal constant $\epsilon$, the method has a near-linear runtime of $\mathcal{O}(l_f n_f \log n_f)$. Further details can be found in Appendix A.2.

**Categorical site-rate variation model.** Having paired the sequences, we proceed to estimate site-rates $\alpha_i^f$ and quantized distances $t_j^f$ between pairs. This is done with coordinate ascent until convergence (no change in site-rates) or a maximum of 50 iterations; we find that convergence usually happens after around 10 iterations. We adopt the CAT model of site-rate variation used by FastTree [2], which poses that site-rate $\alpha_i^f$ takes one of $r$ (a hyperparameter) values from a geometric grid between $1/r$ and $r$, i.e., $R = \{r^{-1+2\frac{(r-i)}{(r-1)}} : i = 1, \ldots, r\}$, with a $\Gamma(3, 1/3)$ prior. We use this model of site-rate variation as it is the one used by FastTree, which is one of the fastest phylogenetic tree reconstruction algorithms available and the one used in the original CherryML work [11]; however, our method can be adapted to other models of site-rate variation.

**Site-rate estimation.** To estimate site-rates $\alpha_i^f$ given quantized divergence time $t_j^f$ for each cherry $j$ and the transition-rate matrix $Q$, we perform 0-th order optimization on CherryML's composite likelihood. Each site-rate $\alpha_i^f$ can be optimized independently. To optimize site-rate $\alpha_i^f$, we simply need to find which of the $r$ rates in $R$ maximizes that site's composite likelihood under the Gamma

prior. In other words, letting $\phi(x)$ be the density of the $\Gamma(3, 1/3)$ distribution, we optimize for:

$$\alpha_i^f = \arg\max_{x \in R} \log \phi(x) + \sum_{j=1}^{c_f} \log(e^{x t_j^f Q}[D_f[u_j^f]_i, D_f[v_j^f]_i]) + \log(e^{x t_j^f Q}[D_f[v_j^f]_i, D_f[u_j^f]_i]).$$

By precomputing all the matrix exponentials $\{\exp(x\tau_k Q) : x \in R, 1 \le k \le b\}$ in time $\mathcal{O}(rbs^3)$, the above objective can be evaluated in time $\mathcal{O}(c_f) = \mathcal{O}(n_f)$ and thus optimized by brute-force search in time $\mathcal{O}(rn_f)$. Since there are a total of $l_f$ sites, optimizing the site-rates this way takes time $\mathcal{O}(l_f rn_f)$. This is equal to the size of the MSA $D_f$, multiplied by the number of rate categories $r$. We initialize the site-rates $\alpha_i^f$ using a simple heuristic described in Appendix A.3.

**Branch length estimation.** To estimate the quantized divergence time $t_j^f$ of each pair given the site-rates $\alpha_i^f$ and transition-rate matrix $Q$, we again perform 0-th order optimization on CherryML's composite likelihood. This means we optimize for

$$t_j^f = \arg\max_{x \in \{\tau_k : 1 \le k \le b\}} \sum_{i=1}^{l_f} \log(e^{\alpha_i^f x Q}[D_f[u_j^f]_i, D_f[v_j^f]_i]) + \log(e^{\alpha_i^f x Q}[D_f[v_j^f]_i, D_f[u_j^f]_i]).$$

By using precomputed matrix exponentials, this objective can be evaluated in time $\mathcal{O}(l_f)$ and optimized with brute-force search in time $\mathcal{O}(bl_f)$. We empirically find that the objective function above is (quasi-)concave in $x$, so that we can optimize it with binary search in time $\mathcal{O}(\log(b)l_f)$, which provides a significant speedup. (We could also have done this for optimizing site-rates, but since $r$ is already typically small for the CAT model, we found it an unnecessary optimization.) This way, optimizing all quantized divergence times $t_j^f$ within a given family takes $\mathcal{O}(n_f \log(b)l_f)$ time. This is equal to the size of the MSA $D_f$, multiplied by $\log(b)$.

**Runtime.** As detailed in Appendix A.4, the above FastCherries algorithm is nearly linear in the size of the MSA, and applying it to a real MSA with 453,819 sequences of length 364 took only about 1000 seconds on 1 CPU core. Furthermore, combined with CherryML to estimate a transition-rate matrix, this leads to an end-to-end estimation procedure that is nearly linear in the dataset size. Also, FastCherries uses linear space, and thus is memory-efficient (up to a constant), as detailed in Appendix A.5.

### 3.3 Regularized estimation of site-specific transition-rate matrices under the SiteRM model

We can employ the above algorithm to derive a regularized procedure for estimation under the more richly parameterized SiteRM model. Recall that the SiteRM model uses *family- and site-specific transition-rate matrices* $Q_i^f$. To estimate these matrices, we first apply our FastCherries algorithm to estimate cherries, distances, and site-rates under the LG model. Given these, we simply apply CherryML's transition-rate matrix estimation procedure [11] at each site separately to obtain $Q_i^f$.

**Regularization.** Of course, the model is over-parameterized in the sense that there is usually not enough data to estimate the site-specific rate matrices accurately. Indeed, $Q_i^f$ has 400 parameters and the only information to support its estimation are the $\lfloor n_f/2 \rfloor$ pairs of amino acids at that site. Even with $n_f$ in the millions, most pairs will not contain mutations at the site since pairs are by definition close sequences, and thus small entries in the rate matrix will be susceptible to large errors. We address this by mixing the empirical counts with pseudocounts from a prior. Specifically, let $Q_0$ be the transition-rate matrix used for regularizing the model – we use the LG transition-rate matrix in applications. We mix the empirical count matrices $C_k$ with pseudocounts from the LG model with transition-rate matrix $Q_0$ and site-rates $\alpha_i^f$. Mixing is performed with a regularization coefficient $\lambda \in [0, 1]$. Formally, we take

$$\tilde{C}_k = (1 - \lambda)C_k + \lambda \|C_k\|_1 P_k,$$

where $P_k$ is the $20 \times 20$ pseudocount matrix with $\|P_k\|_1 = 1$ given by $P_k[x, y] = \pi_{Q_0}[x] \exp(\alpha_i^f \tau_k Q_0)[x, y]$, where $\pi_{Q_0}$ is the stationary distribution of $Q_0$. This choice has the desirable property that when $\lambda = 1$, we recover the LG model: $Q_i^f = \alpha_i^f Q_0$ for all $i, f$. In contrast, when $\lambda = 0$, the prior model is ignored completely. We find that $\lambda = 0.5$ works well in practice.

The runtime analysis of the above procedure is provided in Appendix A.4.3. Our method can thus be thought of as *fine-tuning* the LG model to each site of each protein family. This fine-tuning is achieved by mixing empirical counts with pseudocounts from the 'global' LG model.

# 4 Results

## 4.1 Scalable estimation under the LG model

We applied CherryML with FastCherries to the benchmarks from the CherryML paper [11]. These are described in detail in Appendix A.6.

Figure 2a and Figure 2b respectively illustrate the total runtime and accuracy of the end-to-end method for data simulated under the LG model, using a log-log plot. It is important to note that while in the original CherryML paper the ground truth trees were used and thus only runtime and accuracy of the rate matrix estimation step were assessed, we benchmarked the *end-to-end* procedure without access to the ground truth trees. In particular, we iterated the process of tree estimation and rate matrix estimation four times as is typical on real data, starting from the uniform rate matrix. Accuracy is measured via the median relative error of all the off-diagonal entries in the estimated rate matrix.

**Simulations results.** As Figure 2b shows, CherryML with FastCherries is one to two orders of magnitude faster than CherryML with FastTree [2]. Runtimes of the tree estimation step of CherryML with FastCherries are so fast that total runtime for small dataset sizes is dominated by the PyTorch first-order optimizer (whose runtime $\Theta(gbs^3)$ is independent of input data size). Thus, only for larger dataset sizes does the runtime of CherryML with FastCherries see any noticeable increase. In terms of accuracy, Figure 2 (a) shows that CherryML with FastCherries shows a small asymptotic bias of around $2\%$ median relative error. Otherwise, the relative statistical efficiency of CherryML with FastCherries compared to CherryML using ground truth trees or FastTree is around $50\%$, meaning that CherryML with FastCherries requires approximately twice as much data to achieve the same error. This is similar to the relative statistical efficiency of CherryML compared to MLE with EM.

**Real data results.** On real data benchmarks, we observed similar end-to-end speedups of one to two orders of magnitude for CherryML with FastCherries compared to CherryML with FastTree, while obtaining similar likelihoods on held-out data. This is both for the original LG paper's Pfam dataset, shown in Figure 2c,d, as well as for the QMaker [24] datasets, shown in Supplementary Figure S1.

## 4.2 Variant Effect Prediction with the SiteRM model

Here, we summarize the performance of our SiteRM model on the task of variant effect prediction. Prior work on applying phylogenetic models to variant effect prediction is described in Appendix A.1.

**Data.** A notable resource for this purpose is ProteinGym [25], which features dozens of models benchmarked across both deep mutational scanning (DMS) data and human clinical variants. The DMS substitutions benchmark comprises 2.4M variants across 217 DMS assays, while the human clinical substitutions benchmark contains 63k variants across 2,525 proteins. On the DMS substitutions benchmark, model pathogenicity scores are evaluated against experimental measurements, and evaluation is performed using diverse metrics including Spearman correlation, AUC, MCC, NDCG@10, and top 10 recall. The metrics are aggregated over all 217 assays, in such a way to give equal representation to different kinds of DMS assays, which include protein activity, binding, expression, organismal fitness, and thermostability. The clinical benchmark contains binary pathogenic/benign labels, so that performance is evaluated by averaging AUC across all 2,525 proteins.

**Traditional scoring.** Probabilistic models of proteins such as Potts models [12] and protein language models (e.g., ESM [13]) have been used for the task of variant effect prediction. These models provide either a likelihood function $p(x)$ or, in the case of masked language models, a conditional probability distribution $p(x_M|x_{-M})$ where $M$ is the set of masked indices. These models can be used for variant effect prediction by scoring via log-likelihood ratios, as follows. Letting $x^{\mathrm{mut}}$ denote the mutant and $x^{\mathrm{wt}}$ the wild-type sequence, $p(x)$ may be used to score mutants via $score(x^{\mathrm{mut}}, x^{\mathrm{wt}}) = \log \frac{p(x^{\mathrm{mut}})}{p(x^{\mathrm{wt}})}$. In the case of masked language models, conditional likelihoods $p(x_M|x_{-M})$ may be used to score the substitution of $x_M$ by $x'_M$ via $score(x^{\mathrm{mut}}, x^{\mathrm{wt}}) = \log \frac{p(x'_M|x_{-M})}{p(x_M|x_{-M})}$.

**Our new approach.** We take a conceptually different approach to variant effect prediction by leveraging probabilistic models of protein evolution $p(y|x, t)$. We propose to score mutants via

$$score(x^{\mathrm{mut}}, x^{\mathrm{wt}}) = \log \frac{p(x^{\mathrm{mut}}|x^{\mathrm{wt}}, t)}{p(x^{\mathrm{wt}}|x^{\mathrm{wt}}, t)},$$

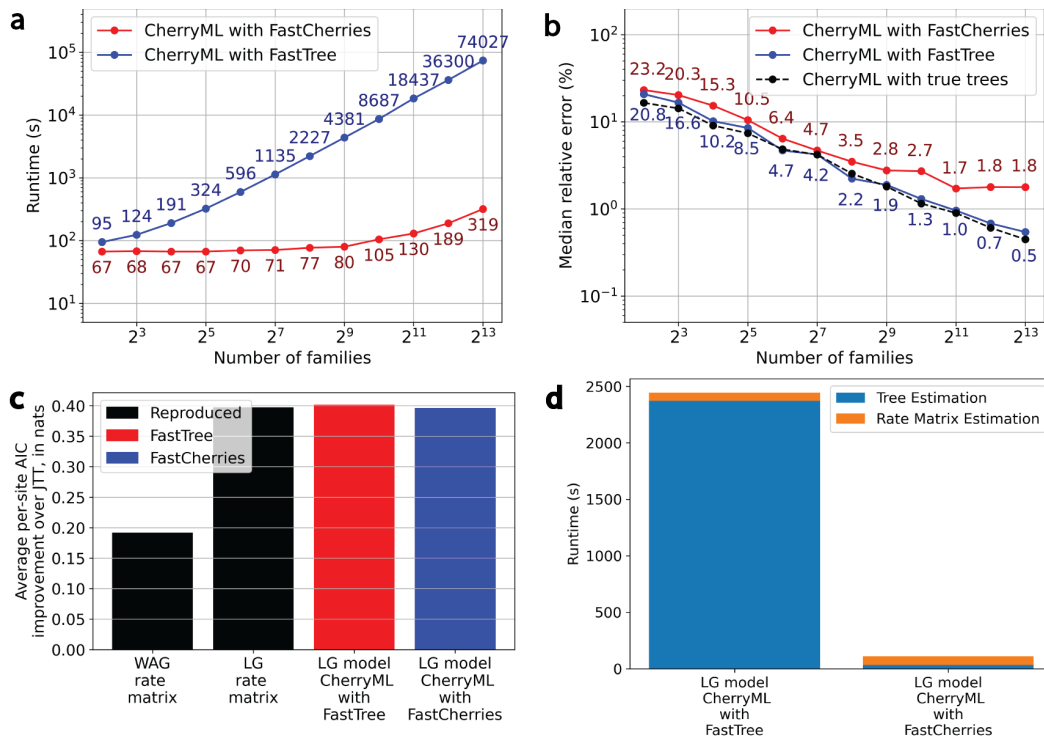

Figure 2: **CherryML with FastCherries applied to the LG model.** (a) End-to-end runtime and (b) median estimation error as a function of sample size for CherryML with FastCherries vs CherryML with FastTree (as well as an oracle with ground truth trees and site rates). Practically, the loss of statistical efficiency for CherryML with FastCherries relative to FastTree or ground truth trees (which perform similarly) is $\approx 50\%$ with a small asymptotic bias of around $2\%$, yet CherryML with FastCherries is two orders of magnitude faster when applied to 1,024 families. The bulk of the end-to-end runtime is taken by rate matrix estimation. The simulation setup is the same as in the CherryML paper [11]. (c) On the benchmark from the LG paper [7], CherryML with FastCherries yields similar likelihood on held-out families compared to CherryML with FastTree, while (d) shows that CherryML with FastCherries is approximately 20 times faster end-to-end than CherryLM with FastTree, with the bottleneck now being rate matrix estimation.

where $t$ is a hyperparameter that controls the size of the evolutionary neighborhood around the wild-type. Our intuition for why this is a good approach is described below.

Consider how the predictions change as $t$ varies. When $t = 0$, we obtain a Dirac delta at $x^{\text{wt}}$, so that *all* mutants get extremely bad scores. As $t$ starts increasing, proteins which are *evolutionarily close* to the wild-type will see an increase in probability. These are proteins that our model thinks would have arisen from the process of mutation and natural selection, making our conditional probability an excellent candidate for variant effect prediction. Finally, when $t = +\infty$, the stationary distribution of the evolutionary model is being used to make predictions. In this case, the wild-type is being *completely* ignored, assuming we are comparing variants with respect to the same wild-type. For classical phylogenetic models, $t$ may be interpreted as the expected number of mutations per site. We use $t = 1$ for variant effect prediction, which means we explore a neighborhood of one expected mutation per site. Note that more conserved sites will have smaller site-rates, so they may not mutate at all in 1 time unit, while other less conserved sites may mutate multiple times.

A key insight is that traditional scoring approaches with $p(x)$ may be thought of as a particular case of our approach with $t = \infty$. Indeed, given an arbitrary model of $p(x)$ one may construct a model of protein evolution $p_{\text{evo}}(y|x, t)$ whose stationary distribution is $p(x)$ via, for example, Metropolis-Hastings, or a continuous-time, discrete diffusion model. This insight has been observed before, most notably the works of EvoVelocity [26] and John Ingraham's Ph.D. thesis [27], which use the energy landscape of protein language models and Potts models respectively to construct an evolutionary model. Unfortunately, likelihoods under this evolutionary model $p_{\text{evo}}(y|x, t)$ will in general be intractable. Nonetheless, the point is that traditional scoring using $p(x^{\text{mut}})$ is then *exactly*

Table 1: Despite being an *independent-sites* model, SiteRM matches or outperforms many notable models, including the large protein language model ESM-1v [13], the alignment-based epistatic models EVmutation [12] and DeepSequence [28], and the inverse-folding model ESM-IF1 [29]. Best performance among these models are shown in boldface.

| Model name | Spearman | AUC | MCC | NDCG | Recall |
|---|---|---|---|---|---|
| SiteRM | **0.423** | **0.734** | **0.330** | **0.776** | 0.207 |
| EVmutation | 0.395 | 0.716 | 0.305 | **0.777** | 0.222 |
| DeepSequence (ensemble) | 0.419 | 0.729 | 0.328 | **0.776** | **0.226** |
| ESM-1v (ensemble) | 0.407 | 0.723 | 0.320 | 0.749 | 0.211 |
| ESM-IF1 | **0.422** | 0.730 | **0.331** | 0.748 | 0.223 |

*equivalent* to scoring with $p_{\text{evo}}(x^{\text{mut}}|x^{\text{wt}}, t = \infty)$. The analogy goes both ways: our approach can be thought of as a particular case of scoring with models of $p(x)$, but where $p(x)$ is constructed by conditioning an evolutionary model around the wild-type. Indeed, letting $p_{\text{evo}}(y|x, t)$ be a model of protein evolution, one may define a model via $p(x) = p_{\text{evo}}(x|x^{\text{wt}}, t)$. In this case, scoring with $p_{\text{evo}}(x|x^{\text{wt}}, t)$ is equivalent to scoring with $p(x)$. This way, the key conceptual difference between our approach and traditional approaches is the act of *conditioning on the wild-type*.

**Example.** We provide an illustrative example that underscores the power of conditioning on the wild-type. Suppose a given site in the MSA for some protein family has four amino acids $I, L, D, E$, each with frequency $25\%$. Note that $I$ and $L$ have similar biochemical properties (they are both non-polar, uncharged, and branched-chain), while $D$ and $E$ are similar (both being negatively charged). Suppose that mutations between $I$ and $L$ or between $D$ and $E$ at this site are tolerated, while other mutations among these residues are deleterious. In this case, using $\log p(x^{\text{mut}})$ may not be able to tell apart the benign mutations at this site from the two pathogenic ones. Indeed, in the extreme case of the site-independent EVmutation model [12], all variants at this site will have exactly the same score. In contrast, our SiteRM model, despite being an independent-sites model, will learn and predict that only mutations that preserve the biochemical properties of the residue will be tolerated. Thus, despite being trained on exactly the same data, our SiteRM model has the built-in capacity to make sensible predictions of this kind whereas EVmutation may not. Although a highly expressive model might be able to succeed at this toy example by leveraging the *correlations* between sites, it is evident that an evolutionary model – even if site-independent – is better placed to leverage insights like this one.

**Variant effect prediction results.** In the case of the SiteRM model with $t = 1$, we use

$$score(x^{\text{mut}}, x^{\text{wt}}) = \log p(x^{\text{mut}}|x^{\text{wt}}, t = 1) = \sum_{i=1}^{l} \log(\exp(Q_i^f)[x_i^{\text{wt}}, x_i^{\text{mut}}]) \qquad (1)$$

to score variants with respect to the same wild-type sequence $x^{\text{wt}}$. Note that $x^{\text{mut}}$ may differ from $x^{\text{wt}}$ at multiple sites, and (1) is an *independent-sites* model, which uses additive per-site effects.

We trained the SiteRM model on the MSAs provided by the ProteinGym benchmark. Duplicate sequences were removed. We used FastCherries with the LG rate matrix and 20 rate categories to estimate cherries and site rates. Then, we use CherryML to estimate site-specific rate matrices, with the LG rate matrix as a prior and with regularization strength $\lambda = 0.5$ such that exactly half the data comes from pseudocounts and the other half from empirical counts. We found that modeling the gap as its own state improved performance, so that our best model uses $21 \times 21$ site-specific rate matrices, with the $21 \times 21$ rate matrix learnt from the LG paper's Pfam dataset as the prior.

Table 1 summarizes the DMS substitution benchmark results for some select notable models. Despite being an independent-sites model, SiteRM outperforms the large protein language model ESM-1v and its inverse-folding version ESM-IF1 on Spearman correlation, AUC, MCC and NDCG. SiteRM also outperforms EVmutation [12], which is based on the seminal Potts model with pairwise interactions, as well as the DeepSequence ensemble model [28]. Only on Recall@10 do we observe less competitive results. We attribute the remarkable performance of our independent-sites model to conditioning on the wild-type, which is absent from competing approaches, and to our ability to estimate family- and site-specific rate matrices without overfitting. Full results with all models reported in the ProteinGym paper [25] are provided in Supplementary Table S1. The table also shows ablations for the SiteRM model where we reduce the size of the training dataset by subsampling sequences in the MSA, or by using FastTree instead of FastCherries, or by excluding gaps. The

largest MSA processed by FastCherries had approximately 454K distinct sequences, each of length 364. FastCherries took approximately 1,000 seconds on a single CPU core to estimate cherries and site rates for this MSA. Of these, 30 seconds were spent on the divide-and-conquer pairing step, and 970 seconds on distance and site rate estimation. Subsequently, $\sim 1$ second was spent per site-specific rate matrix estimation, meaning a total time of around 1,500 seconds. Extrapolating our estimates shown in Supplementary Table S1, FastTree would have taken around 50 times longer.

Finally, on the human clinical benchmark, Supplementary Table S2 shows that SiteRM achieves an AUC of $0.911$, which is $\sim 0.02$ higher than ESM-1b [13], and less than 0.01 below the state-of-the-art.

## 5   Discussion

We have introduced an end-to-end method to estimate amino acid substitution rate matrices from MSAs alone. By combining CherryML [11] with our FastCherries algorithm, it achieves near-linear runtime. Through simulations, we rigorously studied the computational and statistical efficiency of the method, finding it to be around 10 to 100 times faster than CherryML with FastTree [2], all while being only $50\%$ less efficient and having a small asymptotic bias of around $2\%$ median relative error which we find negligible in practical applications. CherryML with FastCherries easily runs on MSAs with hundreds of thousands of sequences. We plan to contribute to the open-source CherryML package with our FastCherries method so that all researchers may use the end-to-end pipeline easily.

By leveraging FastCherries' scalability, we performed regularized inference under the SiteRM model, providing site-specific rate matrices for a given MSA. We applied this *independent-sites* model of protein evolution to the task of variant effect prediction and found that it outperforms many well-established models such as deep protein language models. We believe that this seemingly paradoxical result that an independent-sites model outperforms epistatic models is largely explained by our new conceptual approach, which conditions a model of protein evolution around the wild-type.

**Future directions.** Our work leaves several intriguing avenues for future research. First, although in this work we focused on classical independent-sites models of protein evolution, the large amount of data generated quickly by our FastCherries method may be used to train more complex models of protein evolution that go beyond the independent-sites assumption, such as deep neural networks. These may combine the best of both worlds: the power of modeling the evolutionary process in time and the ability to take into account complex correlations between protein sites. Our work also opens up a new avenue in statistical phylogenetics by enabling tree reconstruction with *site-specific rate matrices*. Software such as IQTree [5] already allows phylogenetic tree reconstruction under a 'partition model', where sites may be grouped such that all sites in the same group evolve under the same transition-rate matrix. Exploring extreme cases of the partition model where each site is its own partition, as in our SiteRM model, is an exciting and new avenue of research that promises to improve phylogenetic tree reconstruction, and find its way into other applications such as ancestral sequence reconstruction. To this end, we plan to release the site-specific rate matrices we have estimated for all 15,051 protein families in the TrRosetta dataset [30]. More broadly, our idea of bypassing tree estimation through the lens of composite likelihood may enable further methodological developments in other related models, such as mixture models [31]. All in all, we expect our work to have a lasting impact in both statistical phylogenetics and computational variant effect prediction.

**Limitations.** One limitation of our current method is that it assumes – just like essentially all of phylogenetics – that the model of protein evolution is time-reversible, thereby constraining the set of possible rate matrices. This may be an important source of model misspecification in some cases, and there is an emerging body of work extending phylogenetic methods to time-irreversible models [32]. Although we find that our method has outstanding performance on VEP, even when compared to full tree reconstruction methods such as FastTree, our method does exhibit a small but non-zero amount of asymptotic bias which may matter in some other downstream applications. For example, the rate matrices estimated with CherryML using FastCherries may yield different, less accurate tree topologies from those obtained with CherryML using FastTree or other tree reconstruction methods (such as PhyML [4]). The extent to which this is true requires further investigation.

## Acknowledgments and Disclosure of Funding

This research is supported in part by NIH grants R56-HG013117 and R01-HG013117. S.P. would like to acknowledge Anastasia Ignatieva for early discussion about the project's idea.

# References

[1] Alexandros Stamatakis. RAxML version 8: a tool for phylogenetic analysis and post-analysis of large phylogenies. *Bioinformatics*, 30(9):1312–1313, 01 2014.

[2] Morgan N. Price, Paramvir S. Dehal, and Adam P. Arkin. Fasttree 2 – approximately maximum-likelihood trees for large alignments. *PLoS ONE*, 5(3):e9490, Mar 2010.

[3] Fredrik Ronquist and John P. Huelsenbeck. MrBayes 3: Bayesian phylogenetic inference under mixed models. *Bioinformatics*, 19(12):1572–1574, 08 2003.

[4] Stéphane Guindon, Jean-François Dufayard, Vincent Lefort, Maria Anisimova, Wim Hordijk, and Olivier Gascuel. New Algorithms and Methods to Estimate Maximum-Likelihood Phylogenies: Assessing the Performance of PhyML 3.0. *Systematic Biology*, 59(3):307–321, 05 2010.

[5] Bui Quang Minh, Heiko A Schmidt, Olga Chernomor, Dominik Schrempf, Michael D Woodhams, Arndt von Haeseler, and Robert Lanfear. IQ-TREE 2: New Models and Efficient Methods for Phylogenetic Inference in the Genomic Era. *Molecular Biology and Evolution*, 37(5):1530–1534, 02 2020.

[6] Emma B. Hodcroft, Nicola De Maio, Rob Lanfear, Duncan R. MacCannell, Bui Quang Minh, Heiko A. Schmidt, Alexandros Stamatakis, Nick Goldman, and Christophe Dessimoz. Want to track pandemic variants faster? Fix the bioinformatics bottleneck. *Nature*, 591(7848):30–33, March 2021.

[7] Si Quang Le and Olivier Gascuel. An Improved General Amino Acid Replacement Matrix. *Molecular Biology and Evolution*, 25(7):1307–1320, 03 2008.

[8] S. Roch. A short proof that phylogenetic tree reconstruction by maximum likelihood is hard. *IEEE/ACM Transactions on Computational Biology and Bioinformatics*, 3(01):92–94, jan 2006.

[9] Peter S. Klosterman, Andrew V. Uzilov, Yuri R. Bendaña, Robert K. Bradley, Sharon Chao, Carolin Kosiol, Nick Goldman, and Ian Holmes. Xrate: a fast prototyping, training and annotation tool for phylo-grammars. *BMC Bioinformatics*, 7(1):428, 2006.

[10] Ian H Holmes. Historian: accurate reconstruction of ancestral sequences and evolutionary rates. *Bioinformatics*, 33(8):1227–1229, 01 2017.

[11] Sebastian Prillo, Yun Deng, Pierre Boyeau, Xingyu Li, Po-Yen Chen, and Yun S. Song. CherryML: scalable maximum likelihood estimation of phylogenetic models. *Nature Methods*, 20(8):1232–1236, August 2023.

[12] Thomas A. Hopf, John B. Ingraham, Frank J. Poelwijk, Charlotta P. I. Schärfe, Michael Springer, Chris Sander, and Debora S. Marks. Mutation effects predicted from sequence co-variation. *Nature Biotechnology*, 35(2):128–135, February 2017.

[13] Joshua Meier, Roshan Rao, Robert Verkuil, Jason Liu, Tom Sercu, and Alex Rives. Language models enable zero-shot prediction of the effects of mutations on protein function. In M. Ranzato, A. Beygelzimer, Y. Dauphin, P.S. Liang, and J. Wortman Vaughan, editors, *Advances in Neural Information Processing Systems*, volume 34, pages 29287–29303. Curran Associates, Inc., 2021.

[14] M. O. Dayhoff and R. M. Schwartz. Chapter 22: A model of evolutionary change in proteins. In *in Atlas of Protein Sequence and Structure*, 1978.

[15] David T. Jones, William R. Taylor, and Janet M. Thornton. The rapid generation of mutation data matrices from protein sequences. *Comput. Appl. Biosci.*, 8(3):275–282, 1992.

[16] Simon Whelan and Nick Goldman. A General Empirical Model of Protein Evolution Derived from Multiple Protein Families Using a Maximum-Likelihood Approach. *Molecular Biology and Evolution*, 18(5):691–699, 05 2001.

[17] Ziheng Yang. Maximum likelihood phylogenetic estimation from dna sequences with variable rates over sites: approximate methods. *Journal of Molecular Evolution*, 39(3):306–314, 1994.

[18] Cristiano Varin, Nancy Reid, and David Firth. An overview of composite likelihood methods. *Statistica Sinica*, 21(1):5–42, 2011.

[19] Adam Paszke, Sam Gross, Soumith Chintala, Gregory Chanan, Edward Yang, Zachary Devito, Zeming Lin, Alban Desmaison, Luca Antiga, and Adam Lerer. Automatic differentiation in pytorch. In *Advances in Neural Information Processing Systems 30*, 2017.

[20] Martín Abadi, Ashish Agarwal, Paul Barham, Eugene Brevdo, Zhifeng Chen, Craig Citro, Greg S. Corrado, Andy Davis, Jeffrey Dean, Matthieu Devin, Sanjay Ghemawat, Ian Goodfellow, Andrew Harp, Geoffrey Irving, Michael Isard, Yangqing Jia, Rafal Jozefowicz, Lukasz Kaiser, Manjunath Kudlur, Josh Levenberg, Dandelion Mané, Rajat Monga, Sherry Moore, Derek Murray, Chris Olah, Mike Schuster, Jonathon Shlens, Benoit Steiner, Ilya Sutskever, Kunal Talwar, Paul Tucker, Vincent Vanhoucke, Vijay Vasudevan, Fernanda Viégas, Oriol Vinyals, Pete Warden, Martin Wattenberg, Martin Wicke, Yuan Yu, and Xiaoqiang Zheng. TensorFlow: Large-scale machine learning on heterogeneous systems, 2015. Software available from tensorflow.org.

[21] Philipp Bader, Sergio Blanes, and Fernando Casas. Computing the matrix exponential with an optimized taylor polynomial approximation. *Mathematics*, 7(12), 2019.

[22] Diederik P Kingma and Jimmy Ba. Adam: A method for stochastic optimization. In *Proceedings of the 3rd International Conference on Learning Representations (ICLR)*, 2015.

[23] Nick Bhattacharya, Neil Thomas, Roshan Rao, Justas Daupras, Peter K Koo, David Baker, Yun S. Song, and Sergey Ovchinnikov. Single layers of attention suffice to predict protein contacts, 2021.

[24] Bui Quang Minh, Cuong Cao Dang, Le Sy Vinh, and Robert Lanfear. QMaker: Fast and Accurate Method to Estimate Empirical Models of Protein Evolution. *Systematic Biology*, 70(5):1046–1060, 02 2021.

[25] Pascal Notin, Aaron Kollasch, Daniel Ritter, Lood van Niekerk, Steffanie Paul, Han Spinner, Nathan Rollins, Ada Shaw, Rose Orenbuch, Ruben Weitzman, Jonathan Frazer, Mafalda Dias, Dinko Franceschi, Yarin Gal, and Debora Marks. Proteingym: Large-scale benchmarks for protein fitness prediction and design. In A. Oh, T. Neumann, A. Globerson, K. Saenko, M. Hardt, and S. Levine, editors, *Advances in Neural Information Processing Systems*, volume 36, pages 64331–64379. Curran Associates, Inc., 2023.

[26] Brian L. Hie, Kevin K. Yang, and Peter S. Kim. Evolutionary velocity with protein language models predicts evolutionary dynamics of diverse proteins. *Cell Systems*, 13(4):274–285.e6, 2022.

[27] Probabilistic Models of Structure in Biological Sequences - ProQuest.

[28] Adam J. Riesselman, John B. Ingraham, and Debora S. Marks. Deep generative models of genetic variation capture the effects of mutations. *Nature Methods*, 15(10):816–822, October 2018.

[29] Chloe Hsu, Robert Verkuil, Jason Liu, Zeming Lin, Brian Hie, Tom Sercu, Adam Lerer, and Alexander Rives. Learning inverse folding from millions of predicted structures. In Kamalika Chaudhuri, Stefanie Jegelka, Le Song, Csaba Szepesvari, Gang Niu, and Sivan Sabato, editors, *Proceedings of the 39th International Conference on Machine Learning*, volume 162 of *Proceedings of Machine Learning Research*, pages 8946–8970. PMLR, 17–23 Jul 2022.

[30] Jianyi Yang, Ivan Anishchenko, Hahnbeom Park, Zhenling Peng, Sergey Ovchinnikov, and David Baker. Improved protein structure prediction using predicted interresidue orientations. *Proceedings of the National Academy of Sciences*, 117(3):1496–1503, 2020.

[31] Si Quang Le, Cuong Cao Dang, and Olivier Gascuel. Modeling Protein Evolution with Several Amino Acid Replacement Matrices Depending on Site Rates. *Molecular Biology and Evolution*, 29(10):2921–2936, 04 2012.

[32] Cuong Cao Dang, Bui Quang Minh, Hanon McShea, Joanna Masel, Jennifer Eleanor James, Le Sy Vinh, and Robert Lanfear. nQMaker: Estimating Time Nonreversible Amino Acid Substitution Models. *Systematic Biology*, 71(5):1110–1123, 02 2022.

[33] Cuong Dang, Le Vinh, Olivier Gascuel, Bart Hazes, and Quang Le. Fastmg: a simple, fast, and accurate maximum likelihood procedure to estimate amino acid replacement rate matrices from large data sets. *BMC bioinformatics*, 15:341, 10 2014.

[34] Naruya Saitou and Masatoshi Nei. The neighbor-joining method: a new method for reconstructing phylogenetic trees. *Molecular biology and evolution*, 4(4):406–425, 1987.

[35] Elodie Laine, Yasaman Karami, and Alessandra Carbone. GEMME: A Simple and Fast Global Epistatic Model Predicting Mutational Effects. *Molecular Biology and Evolution*, 36(11):2604–2619, 08 2019.

[36] Kazutaka Katoh and Hiroyuki Toh. PartTree: an algorithm to build an approximate tree from a large number of unaligned sequences. *Bioinformatics*, 23(3):372–374, 11 2006.

[37] Jung-Eun Shin, Adam J. Riesselman, Aaron W. Kollasch, Conor McMahon, Elana Simon, Chris Sander, Aashish Manglik, Andrew C. Kruse, and Debora S. Marks. Protein design and variant prediction using autoregressive generative models. *Nature Communications*, 12(1):2403, April 2021.

[38] Jonathan Frazer, Pascal Notin, Mafalda Dias, Aidan Gomez, Joseph K. Min, Kelly Brock, Yarin Gal, and Debora S. Marks. Disease variant prediction with deep generative models of evolutionary data. *Nature*, 599(7883):91–95, November 2021.

[39] Ethan C. Alley, Grigory Khimulya, Surojit Biswas, Mohammed AlQuraishi, and George M. Church. Unified rational protein engineering with sequence-based deep representation learning. *Nature Methods*, 16(12):1315–1322, December 2019.

[40] Kevin K. Yang, Nicolo Fusi, and Alex X. Lu. Convolutions are competitive with transformers for protein sequence pretraining. *Cell systems*, 15(3):286–294.e2, 2024.

[41] Daniel Hesslow, Niccoló Zanichelli, Pascal Notin, Iacopo Poli, and Debora Marks. Rita: a study on scaling up generative protein sequence models. *arXiv preprint arXiv:2205.05789*, 2022.

[42] Erik Nijkamp, Jeffrey A. Ruffolo, Eli N. Weinstein, Nikhil Naik, and Ali Madani. Progen2: Exploring the boundaries of protein language models. *Cell systems*, 14(11):968–978.e3, 2023.

[43] Céline Marquet, Michael Heinzinger, Tobias Olenyi, Christian Dallago, Kyra Erckert, Michael Bernhofer, Dmitrii Nechaev, and Burkhard Rost. Embeddings from protein language models predict conservation and variant effects. *Human Genetics*, 141(10):1629–1647, October 2022.

[44] Roshan M Rao, Jason Liu, Robert Verkuil, Joshua Meier, John Canny, Pieter Abbeel, Tom Sercu, and Alexander Rives. Msa transformer. In Marina Meila and Tong Zhang, editors, *Proceedings of the 38th International Conference on Machine Learning*, volume 139 of *Proceedings of Machine Learning Research*, pages 8844–8856. PMLR, 18–24 Jul 2021.

[45] Pascal Notin, Mafalda Dias, Jonathan Frazer, Javier Marchena-Hurtado, Aidan Gomez, Debora S Marks, and Yarin Gal. Tranception: protein fitness prediction with autoregressive transformers and inference-time retrieval. *arXiv (Cornell University)*, 2022.

[46] Pascal Notin, Lood Van Niekerk, Aaron W Kollasch, Daniel Ritter, Yarin Gal, and Debora Susan Marks. TranceptEVE: Combining family-specific and family-agnostic models of protein sequences for improved fitness prediction. In *NeurIPS 2022 Workshop on Learning Meaningful Representations of Life*, 2022.

[47] Kevin K Yang, Niccolò Zanichelli, and Hugh Yeh. Masked inverse folding with sequence transfer for protein representation learning. *Protein Engineering, Design and Selection*, 36:gzad015, 10 2022.

[48] J. Dauparas, I. Anishchenko, N. Bennett, H. Bai, R. J. Ragotte, L. F. Milles, B. I. M. Wicky, A. Courbet, R. J. de Haas, N. Bethel, P. J. Y. Leung, T. F. Huddy, S. Pellock, D. Tischer, F. Chan, B. Koepnick, H. Nguyen, A. Kang, B. Sankaran, A. K. Bera, N. P. King, and D. Baker. Robust deep learning–based protein sequence design using proteinmpnn. *Science*, 378(6615):49–56, 2022.

# A   Appendix

## A.1   Related Work

Speeding up the tree reconstruction step required by rate matrix estimation has been of interest in recent years as genomic datasets have increased in size. In this direction, one work related to ours is that of FastMG [33], which proposes naively splitting the input MSAs into sub-MSAs, and then running an end-to-end estimation procedure using these MSAs. The MSA splitting procedure is performed by using partial tree reconstruction with the Neighbor-Joining (NJ) algorithm [34] applied to the Hamming distance. In this sense, their method has a similar flavor to ours, since we are decomposing each MSA into pairs of sequences. However, there are a number of key differences between our approach and that of FastMG. Firstly, because FastMG proposes to naively split the MSAs into sub-MSAs and treat each independently, FastMG cannot split the MSA into sub-MSAs that are too small. Indeed, it is statistically infeasible to estimate site-rates reliably from too few sequences, since the site-rate at each position is only supported by the few observations at that site. In particular, it is impossible to estimate the site-rates reliably with FastMG by splitting each MSA into pairs of sequences, since only two amino acids would support the rate at that site. Therefore, FastMG typically uses sub-MSAs of size 16. Because of this, FastMG still requires full phylogenetic tree reconstruction (albeit on smaller trees of size 16). A major limitation is that FastMG does not estimate site-rates for the original MSAs – it only does so for the sub-MSAs. This means that estimation under the SiteRM model cannot be naturally performed, since the site-rates are required by the LG prior we use as a regularizer. Our method can thus be thought of as a version of FastMG with *parameter sharing*, wherein sub-MSAs of size 2 share the site-rate parameters, thus allowing us to capture rich site-rate variation reliably and estimate site-rates for the original MSAs. Finally, FastMG uses NJ to decompose the MSA, whose runtime is in general $\mathcal{O}(n^3)$, while we propose to use a near-linear time $\mathcal{O}(n \log n)$ algorithm. Together, these ingredients lead to our near-linear time algorithm with massive scalability.

The application of phylogenetic methods to variant effect prediction is not new. A notable method in this space is GEMME [35], which uses phylogenetic trees and hand-crafted statistics for the purpose of variant effect prediction. This method performs on-par with the state-of-the-art method TranceptEVE for some metrics on the DMS and clinical substitutions benchmark. The main limitation of GEMME is that it is based on hand-crafted features and lacks a probabilistic basis, limiting its ability to scale to larger datasets in a seamless way. We expect that by training more sophisticated, *epistatic* models of protein evolution on the millions of training pairs $(x, y, t)$ generated by our method, one may be able to scale up our approach and further improve on variant effect prediction.

## A.2   Further details of the divide-and-conquer algorithm

Although the divide-and-conquer algorithm described in Section 3.2 does not automatically guarantee balanced splits, they can be achieved by modifying the splitting procedure by first sorting the points based on the ratio $d(z, z_1)/d(z, z_2) \in [0, +\infty]$, assigning the $\Theta(\epsilon)$ ones with smallest ratio to $S_1$ and the $\Theta(\epsilon)$ ones with greatest ratio to $S_2$; the remaining points are assigned with the original procedure. Moreover, this modified procedure only needs to be applied if the original split is unbalanced to begin with. Empirically, enforcing theoretically balanced splits is completely unnecessary, and we use the original simple splitting criteria, which empirically achieves near-linear runtime on real data inputs. A similar algorithm called PartTree [36] has been proposed in the past for scalable phylogenetic tree reconstruction, but it has seen little adoption due to its inferior topological accuracy compared to methods [2]. It is important to note that we care about rate matrix estimation, *not* tree estimation (since the trees $\mathcal{T}$ are nuisance parameters in our model), which is why such near-linear time algorithm turns out to be profitable in our setting. As we demonstrate in the Results section, the tree estimates (specifically, *cherries*) obtained this way are good enough for our purposes.

## A.3   Site-rate initialization

To initialize the site-rates $\alpha_i^f$ in our FastCherries method, we use the following heuristic that tries to align the initial site-rates with the $\Gamma(3, 1/3)$ prior. First, for each site $i$, we count the number of pairs

$(x, y)$ supporting a mutation at that site:

$$\text{muts}_i^f = \sum_{1 < j < k < n_f} 1\{D_{ji}^f \neq D_{ki}^f\}$$

which can be easily computed in time $\mathcal{O}(n_f l_f + l_f s)$ by just counting the number of occurrences of each state at each site. Gaps are treated as missing data and are ignored when computing the above quantity. Next, we sort the sites based on the number of such mutations in time $\mathcal{O}(l_f \log l_f)$. Recall that the site-rates in the CAT model with $r$ rate categories are geometrically spaced between $1/r$ and $r$. Furthermore, a $\Gamma(3, 1/3)$ prior is used. Call these rates $\gamma_1, \ldots, \gamma_r$. Using our sorted order for sites based on number of mutations, we assign a number of sites to rate category $\gamma_i$ proportional to the measure of the $\Gamma(3, 1/3)$ distribution between $\sqrt{\gamma_{i-1}\gamma_i}$ and $\sqrt{\gamma_i\gamma_{i+1}}$, where $\gamma_0 = 0$ and $\gamma_{r+1} = +\infty$.

## A.4 Runtime

### A.4.1 Runtime of the FastCherries algorithm

Letting $a$ be the number of coordinate ascent steps used for optimizing site-rates and divergence times, the runtime of our FastCherries algorithm described in Section 3.2 for family $f$ is

$$\mathcal{O}(\underbrace{l_f n_f \log n_f}_{\substack{\text{Pairing} \\ \text{step}}} + \underbrace{rbs^3}_{\substack{\text{Matrix} \\ \text{exponential} \\ \text{precomputation}}} + \underbrace{n_f l_f + l_f s + l_f \log l_f}_{\substack{\text{Site} \\ \text{rate} \\ \text{initialization}}} + \underbrace{a}_{\substack{\text{Coordinate} \\ \text{ascent}}} (\underbrace{l_f r n_f}_{\substack{\text{Site} \\ \text{rate} \\ \text{estimation}}} + \underbrace{n_f \log(b) l_f}_{\substack{\text{Distance} \\ \text{estimation}}})),$$

Which is nearly linear in $n_f l_f$, the size of the MSA $D_f$. Large applications encountered in practice have $n_f = 450,000$, $l_f = 300$, $r = 20$, $b = 100$, $a = 10$, $s = 20$. For a real MSA with over 450,000 sequences of length 364, we find that our method takes around 1000 seconds on 1 CPU core.

### A.4.2 End-to-end runtime

Finally, combined with CherryML's procedure to estimate a transition-rate matrix $Q$ given pairs $(u_j^f, v_j^f)$, divergences $t_j^f$ and site-rates $\alpha_i^f$, we obtain an end-to-end procedure for estimating a rate matrix $Q$ under the LG model. Assuming for simplicity as in CherryML's analysis that all families have the same number of sequences $n_f = n$ and lengths $l_f = l$, denoting by $g$ the number of first-order gradient descent steps of CherryML (implemented in PyTorch using the Adam optimizer), $q$ the number of iterations of the whole process (outer coordinate ascent), and recalling that $m$ is the number of families, we obtain an end-to-end runtime of

$$\mathcal{O}(\underbrace{q}_{\substack{\text{Outer} \\ \text{coordinate} \\ \text{ascent}}} [\underbrace{mnl \log n}_{\substack{\text{Pairing} \\ \text{step}}} + \underbrace{rbs^3}_{\substack{\text{Matrix} \\ \text{exponential} \\ \text{precomputation}}} + \underbrace{mnl + mls + ml \log l}_{\substack{\text{Site} \\ \text{rate} \\ \text{initialization}}} +$$

$$\underbrace{a}_{\substack{\text{Inner} \\ \text{coordinate} \\ \text{ascent}}} (\underbrace{mnlr}_{\substack{\text{Site} \\ \text{rate} \\ \text{estimation}}} + \underbrace{mnl \log(b)}_{\substack{\text{Distance} \\ \text{estimation}}}) + \underbrace{mnl \log b}_{\substack{\text{CherryML} \\ \text{counting} \\ \text{step}}} + \underbrace{gbs^3}_{\substack{\text{CherryML} \\ \text{first-order} \\ \text{optimizer} \\ \text{(PyTorch)}}} ]).$$

In practice, as in the CherryML work, $q = 4$, $g = 2000$. Thus, this end-to-end runtime is nearly linear in the dataset size $mnl$ up to a logarithmic factor $q \log n$ and a constant factor $qa(r + \log(b))$. The procedure is also embarrassingly parallel over families $f$.

### A.4.3 Runtime for estimating site-specific transition-rate matrices

Empirically, on a large MSA with around 450,000 sequences of length 364, it takes around 1000 seconds to estimate branch length and site-rates, and thereafter, it takes around 1 second per site to estimate a site-specific transition-rate matrix $Q_i^f$ using the method described in Section 3.3. Thus, for an MSA with 450,000 sequences of length 364, it takes approximately 1500 seconds end-to-end to estimate site-specific rate matrices. We should remark that unlike in the original CherryML procedure, we (1) initialize the optimizer with $\alpha_i^f Q_0$ instead of their JTT-IPW strategy [11], and (2) use only 100 steps of the Adam optimizer [22] instead of 2000. Indeed, since we are regularizing with $\alpha_i^f Q_0$ this is a reasonable initialization, and further, we find that 2000 steps is unnecessarily large, with

convergence already happening at around 100 steps. This observation is helpful to further boost scalability.

### A.5  Note on space complexity

Our method FastCherries uses linear space. Briefly, the logarithmic factors in the computational runtime come from divide-and-conquer and binary searches, which do not translate to the space complexity. In more detail, the space complexity when processing the $m$ MSAs in parallel using $p$ processes is $\mathcal{O}(ps^2br + pnl + pls)$. Here, $\mathcal{O}(ps^2br)$ is the cost of the precomputed matrix exponentials, which is shared across all MSAs (and can also be shared across all processes, in which case the space drops to $\mathcal{O}(s^2br)$). The second term $\mathcal{O}(pnl)$ is the cost to load the $p$ MSAs into memory. For each MSA, pairing takes $\mathcal{O}(n)$ space, since it takes $\mathcal{O}(n)$ space to store the distance between the pivot and all other sequences. The space for computing the initial site rates for an MSA is $\mathcal{O}(ls + r)$. Here, the $\mathcal{O}(ls)$ term comes from the count matrix $\texttt{count}[i, j]$ which tracks the number of times that character $j$ occurs at position $i$. Computing site rates given branch lengths for an MSA takes $\mathcal{O}(r)$ space, and computing branch lengths given site rates takes $\mathcal{O}(l)$ space. In both cases, we only track the best rate for the current site or best branch length for the current site. This way, our method FastCherries is space efficient (up to a constant).

### A.6  CherryML benchmark details

There are two kinds of benchmarks in the CherryML work [11] which we leverage: a simulated data benchmark and a real data benchmark. In the simulated data benchmark, a ground truth rate matrix (the LG rate matrix) is used to simulate MSAs from ground truth trees and site rates. Runtime and accuracy of rate matrix estimation methods are evaluated as a function of the number of MSAs simulated, thus providing insight into the computational runtime and the statistical efficiency of each method. The real data benchmark is a compilation of benchmarks originally reported in the LG paper [7] and in the QMaker paper [24]. In these benchmarks, real MSAs are used to estimate a rate matrix. The quality of the estimated rate matrix is judged by the log-likelihood of MSAs from held-out families, thus probing the quality of the method in real-life settings. Importantly, these real data benchmarks contain MSAs from diverse areas of life (mammalian MSAs, bird MSAs, insect MSAs, and plant MSAs), thus probing the generalization capabilities of the method to different real biological settings.

### A.7  QMaker datasets benchmark

The QMaker work [24] uses diverse datasets to highlight the importance of learning rate matrices which are specific to different domains of life. The CherryML work [11] uses these datasets to showcase the ability of the CherryML method to generalize to these diverse datasets. Here, we use these datasets in the same way to highlight the ability of CherryML with FastCherries to achieve comparable performance to CherryML with FastTree; in the CherryML paper, it was shown that CherryML with FastTree matches the performance of EM with FastTree. Supplementary Figure S1 shows the results, with log-likelihoods on the left and runtimes on the right. We can see that indeed CherryML with FastCherries achieves comparable performance to CherryML with FastTree, all while being 10-100 times faster end-to-end.

The details for this benchmark are as follows: starting from the $LG$ rate matrix, tree estimation (with either FastTree or FastCherries) and rate matrix estimation with CherryML were iterated 4 times. The number of rate categories used is $4$ as in the CherryML paper and the original LG paper [7]. The learnt rate matrices are then evaluated by computing the log-likelihood of the MLE trees fitted using FastTree with $4$ rate categories.

It should be noted that the $\approx 1000$ MSAs in each of these datasets are relatively small, with less than 100 sequences each, which is why FastCherries is so fast and the runtime is barely noticeable. Most of the runtime is spent on CherryML's rate matrix estimator which is implemented in PyTorch [19] using 2000 epochs of the Adam [22] optimizer. In our experience, this is more than enough to achieve convergence, and why we lower it to 100 epochs when training the SiteRM model.

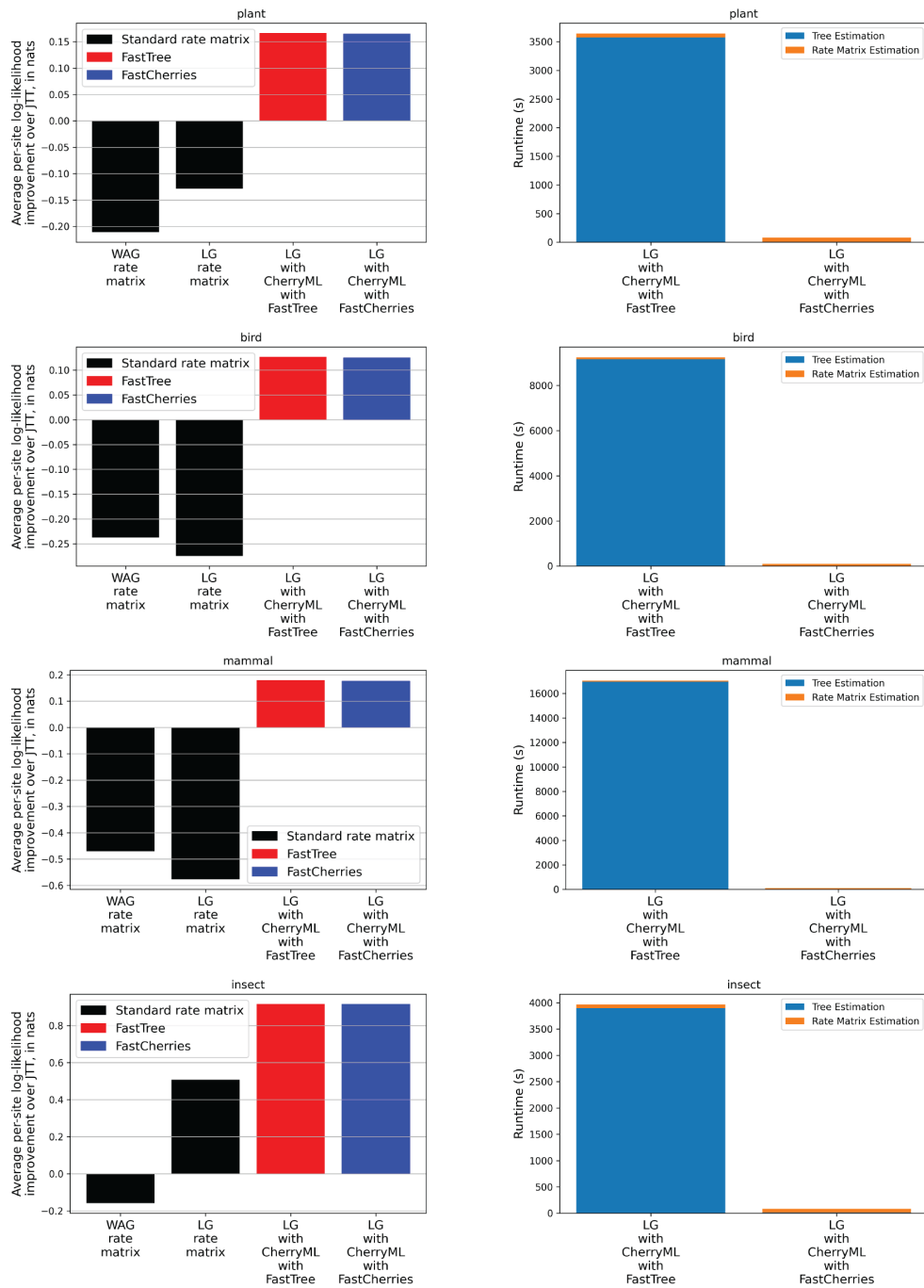

Supplementary Figure S1: **CherryML with FastCherries matches the performance of CherryML with FastTree on diverse datasets.** We reproduced the QMaker results from the CherryML paper [11] and added our CherryML with FastCherries method. We observe similar held-out likelihood at a fraction of end-to-end runtime, which is about 10-100 times faster.

## A.8 Full DMS substitutions results table

Here we provide the full table of results containing all the models reported in the ProteinGym benchmark [25], with our SiteRM model added.

Supplementary Table S1: ProteinGym DMS substitutions benchmark results, with our SiteRM model and ablations added under the 'Protein Evolution, Classical' model type. The ablations explore using less data by subsampling the MSAs, or using FastTree instead of FastCherries, or not treating gaps as a state. We also benchmark the LG model, which uses a global matrix and does quite well already. Despite being an *independent-sites* model, SiteRM outperforms all protein language models expect VESPA, on all metrics except occasionally Recall@10. The approximate CPU time in seconds required for tree/cherry estimation on the largest MSA is shown in the last column 'slowest tree'. Empirically, FastCherries is around 50 times faster than FastTree. Estimating site-specific rate matrices takes around 1 additional second per site.

| Model type | Model name | Spearman | AUC | MCC | NDCG | Recall | Slowest tree |
|---|---|---|---|---|---|---|---|
| Protein Evolution, Classical | LG, FastCherries 1M | 0.377 | 0.710 | 0.297 | 0.763 | 0.185 | 1,000 sec |
| | SiteRM, FastTree 1K | 0.397 | 0.721 | 0.311 | 0.774 | 0.203 | 350 sec |
| | SiteRM, FastCherries 1K | 0.398 | 0.721 | 0.312 | 0.772 | 0.205 | 8 sec |
| | SiteRM, FastTree 10K | 0.404 | 0.724 | 0.315 | 0.776 | 0.204 | 3,100 sec |
| | SiteRM, FastCherries 10K | 0.408 | 0.726 | 0.319 | 0.777 | 0.206 | 60 sec |
| | SiteRM, FastCherries 1M | 0.412 | 0.728 | 0.321 | 0.777 | 0.207 | 1,000 sec |
| | SiteRM, FastCherries 1M, w/gaps | 0.423 | 0.734 | 0.330 | 0.776 | 0.207 | 1,000 sec |
| Alignment-based | Site-Independent [12] | 0.359 | 0.696 | 0.286 | 0.747 | 0.201 | |
| | WaveNet [37] | 0.373 | 0.707 | 0.294 | 0.761 | 0.203 | |
| | EVmutation [12] | 0.395 | 0.716 | 0.305 | 0.777 | 0.222 | |
| | DeepSequence (ensemble) [28] | 0.419 | 0.729 | 0.328 | 0.776 | 0.226 | |
| | EVE (ensemble) [38] | 0.439 | 0.741 | 0.342 | 0.783 | **0.230** | |
| | GEMME [35] | **0.455** | 0.749 | 0.352 | 0.777 | 0.211 | |
| Protein language | UniRep [39] | 0.190 | 0.605 | 0.147 | 0.647 | 0.139 | |
| | CARP (640M) [40] | 0.368 | 0.701 | 0.285 | 0.748 | 0.208 | |
| | ESM-1b [13] | 0.394 | 0.719 | 0.311 | 0.747 | 0.203 | |
| | ESM-2 (15B) [13] | 0.401 | 0.720 | 0.314 | 0.759 | 0.208 | |
| | RITA XL [41] | 0.372 | 0.707 | 0.293 | 0.751 | 0.193 | |
| | ESM-1v (ensemble) [13] | 0.407 | 0.723 | 0.320 | 0.749 | 0.211 | |
| | ProGen2 XL [42] | 0.391 | 0.717 | 0.306 | 0.767 | 0.199 | |
| | VESPA [43] | 0.436 | 0.742 | 0.346 | 0.775 | 0.201 | |
| Hybrid | UniRep evotuned [39] | 0.347 | 0.693 | 0.274 | 0.739 | 0.181 | |
| | MSA Transformer (ensemble) [44] | 0.434 | 0.738 | 0.340 | 0.779 | 0.224 | |
| | Tranception L [45] | 0.434 | 0.739 | 0.341 | 0.779 | 0.220 | |
| | TranceptEVE L [46] | **0.456** | **0.751** | **0.356** | **0.786** | **0.230** | |
| Inverse Folding | ESM-IF1 [29] | 0.422 | 0.730 | 0.331 | 0.748 | 0.223 | |
| | MIF-ST [47] | 0.401 | 0.718 | 0.311 | 0.766 | 0.226 | |
| | ProteinMPNN [48] | 0.258 | 0.639 | 0.196 | 0.713 | 0.186 | |

## A.9 Clinical substitutions results table

Supplementary Table S2: Our *independent-sites* SiteRM model outperforms the large protein language model ESM-1b at human clinical substitutions variant effect prediction (VEP) by nearly 0.02 AUC points, and is less than 0.01 AUC points away from state-of-the-art methods. We again find that modeling gaps as a state in SiteRM (thus learning $21 \times 21$ transition matrices) helps.

| Model name | Mean AUC |
|------------|----------|
| TranceptEVE [46] | 0.920 |
| GEMME [35] | 0.919 |
| EVE [38] | 0.917 |
| SiteRM | 0.911 |
| SiteRM w/o gaps | 0.903 |
| ESM-1b [13] | 0.892 |

## A.10 Compute Resources

The experiments in section 'Scalable estimation under the LG model' were performed on a 'MacBook Pro' model 'MacBookPro18,2' with a 'Apple M1 Max' chip with 10 CPU cores and 32 GB RAM. The ProteinGym variant effect prediction benchmarks were performed on a machine with 32 CPU cores, model 'Intel Xeon Skylake 6230 @ 2.1 GHz', and with 384 GB RAM; however, this is more RAM than necessary and the experiments can likely be performed comfortably with much less RAM. The total compute required by the project was not much larger than that of the experiments reported in the paper, which was dominated by variant effect prediction on human clinical substitutions, which took around 1 day when parallelized with 32 cores and was done last. The large runtime is due to the large number (2525) of clinical MSAs.

